# PAC-Bayesian Generic Chaining

**Jean-Yves Audibert** *

Université Paris 6
Laboratoire de Probabilités et Modèles aléatoires
175 rue du Chevaleret
75013 Paris - France
jyaudibe@ccr.jussieu.fr

**Olivier Bousquet**

Max Planck Institute for Biological Cybernetics
Spemannstrasse 38
D-72076 Tübingen - Germany
olivier.bousquet@tuebingen.mpg.de

## Abstract

There exist many different generalization error bounds for classification. Each of these bounds contains an improvement over the others for certain situations. Our goal is to combine these different improvements into a single bound. In particular we combine the PAC-Bayes approach introduced by McAllester [1], which is interesting for averaging classifiers, with the optimal union bound provided by the generic chaining technique developed by Fernique and Talagrand [2]. This combination is quite natural since the generic chaining is based on the notion of majorizing measures, which can be considered as priors on the set of classifiers, and such priors also arise in the PAC-bayesian setting.

## 1   Introduction

Since the first results of Vapnik and Chervonenkis on uniform laws of large numbers for classes of $\{0, 1\}$-valued functions, there has been a considerable amount of work aiming at obtaining generalizations and refinements of these bounds. This work has been carried out by different communities. On the one hand, people developing empirical processes theory like Dudley and Talagrand (among others) obtained very interesting results concerning the behaviour of the suprema of empirical processes. On the other hand, people exploring learning theory tried to obtain refinements for specific algorithms with an emphasis on data-dependent bounds.

One crucial aspect of all the generalization error bounds is that they aim at controlling the behaviour of the function that is returned by the algorithm. This function is data-dependent and thus unknown before seeing the data. As a consequence, if one wants to make statements about its behaviour (e.g. the difference between its empirical error and true error), one has to be able to *predict* which function is likely to be chosen by the algorithm. But

since this cannot be done exactly, there is a need to provide guarantees that hold simultaneously for several candidate functions. This is known as the union bound. The way to perform this union bound optimally is now well mastered in the empirical processes community.

In the learning theory setting, one is interested in bounds that are as algorithm and data dependent as possible. This particular focus has made concentration inequalities (see e.g. [3]) popular as they allow to obtain data-dependent results in an effortless way. Another aspect that is of interest for learning is the case where the classifiers are randomized or averaged. McAllester [1, 4] has proposed a new type of bound that takes the randomization into account in a clever way.

Our goal is to combine several of these improvements, bringing together the power of the majorizing measures as an optimal union bound technique and the power of the PAC-Bayesian bounds that handle randomized predictions efficiently, and obtain a generalization of both that is suited for learning applications.

The paper is structured as follows. Next section introduces the notation and reviews the previous improved bounds that have been proposed. Then we give our main result and discuss its applications, showing in particular how to recover previously known results. Finally we give the proof of the presented results.

## 2   Previous results

We first introduce the notation and then give an overview of existing generalization error bounds. We consider an input space $\mathcal{X}$, an output space $\mathcal{Y}$ and a probability distribution $P$ on the product space $\mathcal{Z} \triangleq \mathcal{X} \times \mathcal{Y}$. Let $Z \triangleq (X, Y)$ denote a pair of random variables distributed according to $P$ and for a given integer $n$, let $Z_1, \ldots, Z_n$ and $Z'_1, \ldots, Z'_n$ be two independent samples of $n$ independent copies of $Z$. We denote by $P_n$, $P'_n$ and $P_{2n}$ the empirical measures associated respectively to the first, the second and the union of both samples.

To each function $g : \mathcal{X} \to \mathcal{Y}$ we associate the corresponding loss function $f : \mathcal{Z} \to \mathbb{R}$ defined by $f(z) = L[g(x), y]$ where $L$ is a loss function. In classification, the loss function is $L = \mathbb{I}_{g(x) \neq y}$ where $\mathbb{I}$ denotes the indicator function. $\mathcal{F}$ will denote a set of such functions. For such functions, we denote their expectation under $P$ by $Pf$ and their empirical expectation by $P_n f$ (i.e. $P_n f = n^{-1} \sum_{i=1}^{n} f(Z_i)$). $\mathbb{E}_n$, $\mathbb{E}'_n$ and $\mathbb{E}_{2n}$ denote the expectation with respect to the first, second and union of both training samples.

We consider the pseudo-distances $d^2(f_1, f_2) = P(f_1 - f_2)^2$ and similarly $d_n, d'_n$ and $d_{2n}$. We define the covering number $N(\mathcal{F}, \epsilon, d)$ as the minimum number of balls of radius $\epsilon$ needed to cover $\mathcal{F}$ in the pseudo-distance $d$.

We denote by $\rho$ and $\pi$ two probability measures on the space $\mathcal{F}$, so that $\rho Pf$ will actually mean the expectation of $Pf$ when $f$ is sampled according to the probability measure $\rho$. For two such measures, $K(\rho, \pi)$ will denote their Kullback-Leibler divergence ($K(\rho, \pi) = \rho \log \frac{d\rho}{d\pi}$ when $\rho$ is absolutely continuous with respect to $\pi$ and $K(\rho, \pi) = +\infty$ otherwise). Also, $\beta$ denotes some positive real number while $C$ is some positive constant (whose value may differ from line to line) and $\mathcal{M}_+^1(\mathcal{F})$ is the set of probability measures on $\mathcal{F}$. We assume that the functions in $\mathcal{F}$ have range in $[a, b]$.

Generalization error bounds give an upper bound on the difference between the true and empirical error of functions in a given class, which holds with high probability with respect to the sampling of the training set.

**Single function.** By Hoeffding's inequality one easily gets that for each fixed $f \in \mathcal{F}$, with probability at least $1 - \beta$,

$$Pf - P_n f \leq C \sqrt{\frac{\log 1/\beta}{n}} . \tag{1}$$

**Finite union bound.** It is easy to convert the above statement into one which is valid

simultaneously for a finite set of functions $\mathcal{F}$. The simplest form of the union bound gives that with probability at least $1 - \beta$,

$$\forall f \in \mathcal{F}, \; Pf - P_n f \leq C \sqrt{\frac{\log |\mathcal{F}| + \log 1/\beta}{n}} \, . \tag{2}$$

**Symmetrization.** When $\mathcal{F}$ is infinite, the trick is to introduce the second sample $Z_1', \ldots, Z_n'$ and to consider the set of vectors formed by the values of each function in $\mathcal{F}$ on the double sample. When the functions have values in $\{0, 1\}$, this is a finite set and the above union bound applies. This idea was first used by Vapnik and Chervonenkis [5] to obtain that with probability at least $1 - \beta$,

$$\forall f \in \mathcal{F}, \; Pf - P_n f \leq C \sqrt{\frac{\log \mathbb{E}_{2n} N(\mathcal{F}, 1/n, d_{2n}) + \log 1/\beta}{n}} \, . \tag{3}$$

**Weighted union bound and localization.** The finite union bound can be directly extended to the countable case by introducing a probability distribution $\pi$ over $\mathcal{F}$ which weights each function and gives that with probability at least $1 - \beta$,

$$\forall f \in \mathcal{F}, \; Pf - P_n f \leq C \sqrt{\frac{\log 1/\pi(f) + \log 1/\beta}{n}} \, . \tag{4}$$

It is interesting to notice that now the bound depends on the actual function $f$ being considered and not just on the set $\mathcal{F}$. This can thus be called a *localized* bound.

**Variance.** Since the deviations between $Pf$ and $P_n f$ for a given function $f$ actually depend on its variance (which is upper bounded by $Pf^2/n$ or $Pf/n$ when the functions are in $[0, 1]$), one can refine (1) into

$$Pf - P_n f \leq C \left( \sqrt{\frac{Pf^2 \log 1/\beta}{n}} + \frac{\log 1/\beta}{n} \right) , \tag{5}$$

and combine this improvement with the above union bounds. This was done by Vapnik and Chervonenkis [5] (for functions in $\{0, 1\}$).

**Averaging.** Consider a probability distribution $\rho$ defined on a countable $\mathcal{F}$, take the expectation of (4) with respect to $\rho$ and use Jensen's inequality. This gives with probability at least $1 - \beta$,

$$\forall \rho, \; \rho(Pf - P_n f) \leq C \sqrt{\frac{K(\rho, \pi) + H(\rho) + \log 1/\beta}{n}} \, ,$$

where $H(\rho)$ is the Shannon entropy. The l.h.s. is the difference between true and empirical error of a randomized classifier which uses $\rho$ as weights for choosing the decision function (independently of the data). The PAC-Bayes bound [1] is a refined version of the above bound since it has the form (for possibly uncountable $\mathcal{F}$)

$$\forall \rho, \; \rho(Pf - P_n f) \leq C \sqrt{\frac{K(\rho, \pi) + \log n + \log 1/\beta}{n}} \, . \tag{6}$$

To some extent, one can consider that the PAC-Bayes bound is a refined union bound where the gain happens when $\rho$ is not concentrated on a single function (or more precisely $\rho$ has entropy larger than $\log n$).

**Rademacher averages.** The quantity $\mathbb{E}_n \mathbb{E}_\sigma \sup_{f \in \mathcal{F}} \frac{1}{n} \sum \sigma_i f(Z_i)$, where the $\sigma_i$ are independent random signs ($+1, -1$ with probability $1/2$), called the Rademacher average for $\mathcal{F}$, is, up to a constant equal to $\mathbb{E}_n \sup_{f \in \mathcal{F}} Pf - P_n f$ which means that it best captures the complexity of $\mathcal{F}$. One has with probability $1 - \beta$,

$$\forall f \in \mathcal{F}, \; Pf - P_n f \leq C \left( \frac{1}{n} \mathbb{E}_n \mathbb{E}_\sigma \sup_{f \in \mathcal{F}} \sum \sigma_i f(Z_i) + \sqrt{\frac{\log 1/\beta}{n}} \right) . \tag{7}$$

**Chaining.** Another direction in which the union bound can be refined is by considering finite covers of the set of function at different scales. This is called the *chaining* technique, pioneered by Dudley (see e.g. [6]) since one constructs a chain of functions that approximate a given function more and more closely. The results involve the Koltchinskii-Pollard entropy integral as, for example in [7], with probability $1 - \beta$,

$$\forall f \in \mathcal{F}, \ Pf - P_n f \leq C \left( \frac{1}{\sqrt{n}} \mathbb{E}_n \int_0^\infty \sqrt{\log N(\mathcal{F}, \epsilon, d_n)} d\epsilon + \sqrt{\frac{\log 1/\beta}{n}} \right). \quad (8)$$

**Generic chaining.** It has been noticed by Fernique and Talagrand that it is possible to capture the complexity in a better way than using minimal covers by considering majorizing measures (essentially optimal for Gaussian processes). Let $r > 0$ and $(\mathcal{A}_j)_{j \geq 1}$ be partitions of $\mathcal{F}$ of diameter $r^{-j}$ w.r.t. the distance $d_n$ such that $\mathcal{A}_{j+1}$ refines $\mathcal{A}_j$. Using (7) and techniques from [2] we obtain that with probability $1 - \beta, \forall f \in \mathcal{F}$

$$Pf - P_n f \leq C \left( \frac{1}{\sqrt{n}} \mathbb{E}_n \inf_{\pi \in \mathcal{M}_+^1(\mathcal{F})} \sup_{f \in \mathcal{F}} \sum_{j=1}^\infty r^{-j} \sqrt{\log 1/\pi A_j(f)} + \sqrt{\frac{\log 1/\beta}{n}} \right). \quad (9)$$

If one takes partitions induced by minimal covers of $\mathcal{F}$ at radii $r^{-j}$, one recovers (8) up to a constant.

**Concentration.** Using concentration inequalities as in [3] for example, one can get rid of the expectation appearing in the r.h.s. of (3), (8), (7) or (9) and thus obtain a bound that can be computed from the data.

Refining the bound (7) is possible as one can localize it (see e.g. [8]) by computing the Rademacher average only on a small ball around the function of interest. So this comes close to combining all improvements. However it has not been combined with the PAC-Bayes improvement. Our goal is to try and combine all the above improvements.

## 3 Main results

Let $\mathcal{F}$ be as defined in section 2 with $a = 0, b = 1$ and $\pi \in \mathcal{M}_+^1(\mathcal{F})$. Instead of using partitions as in (9) we use approximating sets (which also induce partitions but are easier to handle here). Consider a sequence $S_j$ of embedded finite subsets of $\mathcal{F}$: $\{f_0\} \triangleq S_0 \subset \cdots \subset S_{j-1} \subset S_j \subset \cdots$.

Let $p_j : \mathcal{F} \to S_j$ be maps (which can be thought of as projections) satisfying $p_j(f) = f$ for $f \in S_j$ and $p_{j-1} \circ p_j = p_{j-1}$.

The quantities $\pi$, $S_j$ and $p_j$ are allowed to depend on $X_1^{2n}$ in an exchangeable way (i.e. exchanging $X_i$ and $X_i'$ does not affect their value). For a probability distribution $\rho$ on $\mathcal{F}$, define its $j$-th projection as $\rho_j = \sum_{f \in S_j} \rho\{f' : p_j(f') = f\} \delta_f$, where $\delta_f$ denotes the Dirac measure on $f$. To shorten notations, we denote the average distance between two successive "projections" by $\rho d_j^2 \triangleq \rho d_{2n}^2 [p_j(f), p_{j-1}(f)]$. Finally, let $\Delta_{n,j}(f) \triangleq P_n'[f - p_j(f)] - P_n[f - p_j(f)]$.

**Theorem 1** *If the following condition holds*

$$\lim_{j \to +\infty} \sup_{f \in \mathcal{F}} \Delta_{n,j}(f) = 0, \qquad a.s. \quad (10)$$

*then for any $0 < \beta < 1/2$, with probability at least $1 - \beta$, for any distribution $\rho$, we have*

$$\rho P_n' f - P_n' f_0 \leq \rho P_n f - P_n f_0 + 5 \sum_{j=1}^{+\infty} \sqrt{\frac{\rho d_j^2 K(\rho_j, \pi_j)}{n}} + \frac{1}{\sqrt{n}} \sum_{j=1}^{+\infty} \chi_j(\rho d_j^2),$$

*where $\chi_j(x) = 4\sqrt{x \log\left(4j^2\beta^{-1}\log(e^2/x)\right)}$.*

**Remark 1** *Assumption* (10) *is not very restrictive. For instance, it is satisfied when $\mathcal{F}$ is finite, or when $\lim_{j\to+\infty}\sup_{f\in\mathcal{F}}|f-p_j(f)| = 0$, almost surely or also when the empirical process $\left[f \mapsto Pf - P_nf\right]$ is uniformly continuous (which happens for classes with finite $VC$ dimension in particular) and $\lim_{j\to+\infty}\sup_{f\in\mathcal{F}} d_{2n}(f,p_j(f)) = 0$.*

**Remark 2** *Let $\mathcal{G}$ be a model (i.e. a set of prediction functions). Let $\tilde{g}$ be a reference function (not necessarily in $\mathcal{G}$). Consider the class of functions $\mathcal{F} = \{z \mapsto L[g(x),y] : g \in \mathcal{G} \cup \{\tilde{g}\}\}$. Let $f_0 = L[\tilde{g}(x),y]$. The previous theorem compares the risk on the second sample of any (randomized) estimator with the risk on the second sample of the reference function $\tilde{g}$.*

Now let us give a version of the previous theorem in which the second sample does not appear.

**Theorem 2** *If the following condition holds*

$$\lim_{j\to+\infty}\sup_{f\in\mathcal{F}} \mathbb{E}'_n\left[\Delta_{n,j}(f)\right] = 0, \qquad a.s. \tag{11}$$

*then for any $0 < \beta < 1/2$, with probability at least $1 - \beta$, for any distribution $\rho$, we have*

$$\rho Pf - Pf_0 \leq \rho P_nf - P_nf_0 + 5\sum_{j=1}^{+\infty}\sqrt{\frac{\mathbb{E}'_n[\rho d_j^2]\mathbb{E}'_n[K(\rho_j,\pi_j)]}{n}} + \frac{1}{\sqrt{n}}\sum_{j=1}^{+\infty}\chi_j\left(\mathbb{E}'_n[\rho d_j^2]\right).$$

## 4 Discussion

We now discuss in which sense the result presented above combines several previous improvements in a single bound.
Notice that our bound is localized in the sense that it depends on the function of interest (or rather on the averaging distribution $\rho$) and does not involve a supremum over the class.
Also, the union bound is performed in an optimal way since, if one plugs in a distribution $\rho$ concentrated on a single function, takes a supremum over $\mathcal{F}$ in the r.h.s., and upper bounds the squared distance by the diameter of the partition, one recovers a result similar to (9) up to logarithmic factors but which is localized. Also, when two successive projections are identical, they do not enter in the bound (which comes from the fact that the variance weights the complexity terms). Moreover Theorem 1 also includes the PAC-Bayesian improvement for averaging classifiers since if one considers the set $S_1 = \mathcal{F}$ one recovers a result similar to McAllester's (6) which in addition contains the variance improvement such as in [9].
Finally due to the power of the generic chaining, it is possible to upper bound our result by Rademacher averages, up to logarithmic factors (using the results of [10] and [11]).

As a remark, the choice of the sequence of sets $S_j$ can generally be done by taking successive covers of the hypothesis space with geometrically decreasing radii.

However, the obtained bound is not completely empirical since it involves the expectation with respect to an extra sample. In the transduction setting, this is not an issue, it is even an advantage as one can use the unlabeled data in the computation of the bound. However, in the induction setting, this is a drawback. Future work will focus on using concentration inequalities to give a fully empirical bound.

## 5   Proofs

**Proof of Theorem 1:** The proof is inspired by previous works on PAC-bayesian bounds [12, 13] and on the generic chaining [2]. We first prove the following lemma.

**Lemma 1** *For any $\beta > 0$, $\lambda > 0$, $j \in \mathbb{N}^*$ and any exchangeable function $\pi : \mathcal{X}^{2n} \to \mathcal{M}_+^1(\mathcal{F})$, with probability at least $1 - \beta$, for any probability distribution $\rho \in \mathcal{M}_+^1(\mathcal{F})$, we have*

$$\rho\Big\{ P_n'[p_j(f) - p_{j-1}(f)] - P_n[p_j(f) - p_{j-1}(f)] \Big\}$$
$$\leq \tfrac{2\lambda}{n} \rho d_{2n}^2[p_j(f), p_{j-1}(f)] + \tfrac{K(\rho,\pi) + \log(\beta^{-1})}{\lambda}.$$

**Proof** Let $\lambda > 0$ and let $\pi : \mathcal{X}^{2n} \to \mathcal{M}_+^1(\mathcal{F})$ be an exchangeable function. Introduce the quantity $\Delta_i \triangleq p_j(f)(Z_{n+i}) - p_{j-1}(f)(Z_{n+i}) + p_{j-1}(f)(Z_i) - p_j(f)(Z_i)$ and

$$h \triangleq \lambda P_n'\big[p_j(f) - p_{j-1}(f)\big] - \lambda P_n\big[p_j(f) - p_{j-1}(f)\big] - \frac{2\lambda^2}{n} d_{2n}\big[p_j(f), p_{j-1}(f)\big]. \quad (12)$$

By using the exchangeability of $\pi$, for any $\sigma \in \{-1; +1\}^n$, we have

$$\begin{aligned}
\mathbb{E}_{2n} \pi e^h &= \mathbb{E}_{2n} \pi e^{-\frac{2\lambda^2}{n} d_{2n}[p_j(f), p_{j-1}(f)] + \frac{\lambda}{n} \sum_{i=1}^n \Delta_i} \\
&= \mathbb{E}_{2n} \pi e^{-\frac{2\lambda^2}{n} d_{2n}[p_j(f), p_{j-1}(f)] + \frac{\lambda}{n} \sum_{i=1}^n \sigma_i \Delta_i}.
\end{aligned}$$

Now take the expectation wrt $\sigma$, where $\sigma$ is a $n$-dimensional vector of Rademacher variables. We obtain

$$\begin{aligned}
\mathbb{E}_{2n} \pi e^h &= \mathbb{E}_{2n} \pi e^{-\frac{2\lambda^2}{n} d_{2n}[p_j(f), p_{j-1}(f)]} \prod_{i=1}^n \cosh\left(\tfrac{\lambda}{n} \Delta_i\right) \\
&\leq \mathbb{E}_{2n} \pi e^{-\frac{2\lambda^2}{n} d_{2n}[p_j(f), p_{j-1}(f)]} e^{\sum_{i=1}^n \frac{\lambda^2}{2n^2} \Delta_i^2}
\end{aligned}$$

where at the last step we use that $\cosh s \leq e^{\frac{s^2}{2}}$. Since

$$\Delta_i^2 \leq 2\big[p_j(f)(Z_{n+i}) - p_{j-1}(f)(Z_{n+i})\big]^2 + 2\big[p_j(f)(Z_i) - p_{j-1}(f)(Z_i)\big]^2,$$

we obtain that for any $\lambda > 0$, $\mathbb{E}_{2n} \pi e^h \leq 1$. Therefore, for any $\beta > 0$, we have

$$\mathbb{E}_{2n} \mathbb{I}_{\log \pi e^{h + \log \beta} > 0} = \mathbb{E}_{2n} \mathbb{I}_{\pi e^{h + \log \beta} > 1} \leq \mathbb{E}_{2n} \pi e^{h + \log \beta} \leq \beta, \quad (13)$$

On the event $\big\{ \log \pi e^{h + \log \beta} \leq 0 \big\}$, by the Legendre's transform, for any probability distribution $\rho \in \mathcal{M}_+^1(\mathcal{F})$, we have

$$\rho h + \log \beta \leq \log \pi e^{h + \log \beta} + K(\rho, \pi) \leq K(\rho, \pi), \quad (14)$$

which proves the lemma. ∎

Now let us apply this result to the projected measures $\pi_j$ and $\rho_j$. Since, by definition, $\pi$, $S_j$ and $p_j$ are exchangeable, $\pi_j$ is also exchangeable. Since $p_j(f) = f$ for any $f \in S_j$, with probability at least $1 - \beta$, uniformly in $\rho$, we have

$$\rho_j \Big\{ P_n'[f - p_{j-1}(f)] - P_n[p_j(f) - p_{j-1}(f)] \Big\} \leq \frac{2\lambda}{n} \rho_j d_{2n}^2[f, p_{j-1}(f)] + \frac{K_j'}{\lambda},$$

where $K_j' \triangleq K(\rho_j, \pi_j) + \log(\beta^{-1})$. By definition of $\rho_j$, it implies that

$$\rho \Big\{ P_n'[p_j(f) - p_{j-1}(f)] - P_n[p_j(f) - p_{j-1}(f)] \Big\} \leq \frac{2\lambda}{n} \rho d_{2n}^2[p_j(f), p_{j-1}(f)] + \frac{K_j'}{\lambda}. \quad (15)$$

To shorten notations, define $\rho d_j^2 \triangleq \rho d_{2n}^2[p_j(f), p_{j-1}(f)]$ and $\rho\Delta_j \triangleq \rho\{P_n'[p_j(f) - p_{j-1}(f)] - P_n[p_j(f) - p_{j-1}(f)]\}$. The parameter $\lambda$ minimizing the RHS of the previous equation depends on $\rho$. Therefore, we need to get a version of this inequality which holds uniformly in $\lambda$.

First let us note that when $\rho d_j^2 = 0$, we have $\rho\Delta_j = 0$. When $\rho d_j^2 > 0$, let $m\sqrt{\frac{\log 2}{2n}}$ and $\lambda_k = me^{k/2}$ and let $b$ be a function from $\mathbb{R}^*$ to $(0,1]$ such that $\sum_{k\geq 1} b(\lambda_k) \leq 1$. From the previous lemma and a union bound, we obtain that for any $\beta > 0$ and any integer $j$ with probability at least $1 - \beta$, for any $k \in \mathbb{N}^*$ and any distribution $\rho$, we have

$$\rho\Delta_j \leq \frac{2\lambda_k}{n}\rho d_j^2 + \frac{K(\rho_j, \pi_j) + \log\left([b(\lambda_k)]^{-1}\beta^{-1}\right)}{\lambda_k}.$$

Let us take the function $b$ such that $\left[\lambda \mapsto \frac{\log\left([b(\lambda)]^{-1}\right)}{\lambda}\right]$ is continuous and decreasing. Then there exists a parameter $\lambda^* > 0$ such that $\frac{2\lambda^*}{n}\rho d_j^2 = \frac{K(\rho_j, \pi_j) + \log\left([b(\lambda^*)]^{-1}\beta^{-1}\right)}{\lambda^*}$. For any $\beta < 1/2$, we have $(\lambda^*)^2 \rho d_j^2 \geq \frac{\log 2}{2}n$, hence $\lambda^* \geq m$. So there exists an integer $k \in \mathbb{N}^*$ such that $\lambda_k e^{-1/2} \leq \lambda^* \leq \lambda_k$. Then we have

$$
\begin{aligned}
\rho\Delta_j &\leq \frac{2\lambda^*}{n}\sqrt{e}\rho d_j^2 + \frac{K(\rho_j, \pi_j) + \log\left([b(\lambda_*)]^{-1}\beta^{-1}\right)}{\lambda_*} \\
&= (1 + \sqrt{e})\sqrt{\frac{2}{n}\rho d_j^2\left[K(\rho_j, \pi_j) + \log\left([b(\lambda_*)]^{-1}\beta^{-1}\right)\right]}.
\end{aligned}
\tag{16}
$$

To have an explicit bound, it remains to find an upperbound of $[b(\lambda^*)]^{-1}$. When $b$ is decreasing, this comes down to upperbouding $\lambda^*$. Let us choose $b(\lambda) = \frac{1}{[\log(\frac{e^{2\lambda}}{m})]^2}$ when $\lambda \geq m$ and $b(\lambda) = 1/4$ otherwise. Since $b(\lambda_k) = \frac{4}{(k+4)^2}$, we have $\sum_{k\geq 1} b(\lambda_k) \leq 1$. Tedious computations give $\lambda^* \leq 7m\frac{\sqrt{K_j'}}{\rho d_j^2}$ which combined with (16), yield

$$\rho\Delta_j \leq 5\sqrt{\frac{\rho d_j^2 K(\rho_j, \pi_j)}{n}} + 3.75\sqrt{\frac{\rho d_j^2}{n}\log\left(2\beta^{-1}\log\left[\frac{e^2}{\rho d_j^2}\right]\right)}.$$

By simply using an union bound with weights taken proportional to $1/j^2$, we have that the previous inequation holds uniformly in $j \in \mathbb{N}^*$ provided that $\beta^{-1}$ is replaced with $\frac{\pi^2}{6}j^2\beta^{-1}$ $\left(\text{since } \sum_{j\in\mathbb{N}^*} 1/j^2 = \pi^2/6 \approx 1.64\right)$. Notice that

$$\rho\left[P_n'f - P_n'f_0 + P_nf_0 - P_nf\right] = \rho\Delta_{n,J}(f) + \sum_{j=1}^{J}\rho_j\left[(P_n' - P_n)f - (P_n' - P_n)p_{j-1}(f)\right]$$

because $p_{j-1} = p_{j-1} \circ p_j$. So, with probability at least $1 - \beta$, for any distribution $\rho$, we have

$$
\begin{aligned}
\rho\left[P_n'f - P_n'f_0 + P_nf_0 - P_nf\right] \leq \sup_{\mathcal{F}}\Delta_{n,J} &+ 5\sum_{j=1}^{J}\sqrt{\frac{\rho d_j^2 K(\rho_j, \pi_j)}{n}} \\
&+ 3.75\sum_{j=1}^{J}\sqrt{\frac{\rho d_j^2}{n}\log\left(3.3j^2\beta^{-1}\log\left[\frac{e^2}{\rho d_j^2}\right]\right)}.
\end{aligned}
$$

Making $J \to +\infty$, we obtain theorem 1. $\qquad\square$

**Proof of Theorem 2:** It suffices to modify slightly the proof of theorem 1. Introduce $U \triangleq \sup_\rho\left\{\rho h + \log\beta - K(\rho, \pi)\right\}$, where $h$ is still defined as in equation (12). Inequations (14) implies that $\mathbb{E}_{2n}e^U \leq \beta$. By Jensen's inequality, we get $\mathbb{E}_n e^{\mathbb{E}_n'U} \leq \beta$, hence $\mathbb{E}_n\left\{\mathbb{E}_n'U \geq 0\right\} \leq \beta$. So with probability at least $1 - \beta$, we have $\sup_\rho\mathbb{E}_n'\left\{\rho h + \log\beta - K(\rho, \pi)\right\} \leq \mathbb{E}_n'U \leq 0$. $\qquad\square$

# 6 Conclusion

We have obtained a generalization error bound for randomized classifiers which combines several previous improvements. It contains an optimal union bound, both in the sense of optimally taking into account the metric structure of the set of functions (via the majorizing measure approach) and in the sense of taking into account the averaging distribution. We believe that this is a very natural way of combining these two aspects as the result relies on the comparison of a majorizing measure which can be thought of as a prior probability distribution and a randomization distribution which can be considered as a posterior distribution.

Future work will focus on giving a totally empirical bound (in the induction setting) and investigating possible constructions for the approximating sets $S_j$.

## Footnotes

*Secondary affiliation: CREST, ENSAE, Laboratoire de Finance et Assurance, Malakoff, France

# References

[1] D. A. McAllester. Some PAC-Bayesian theorems. In *Proceedings of the 11th Annual Conference on Computational Learning Theory*, pages 230–234. ACM Press, 1998.

[2] M. Talagrand. Majorizing measures: The generic chaining. *Annals of Probability*, 24(3):1049–1103, 1996.

[3] S. Boucheron, G. Lugosi, and S. Massart. A sharp concentration inequality with applications. *Random Structures and Algorithms*, 16:277–292, 2000.

[4] D. A. McAllester. PAC-Bayesian model averaging. In *Proceedings of the 12th Annual Conference on Computational Learning Theory*. ACM Press, 1999.

[5] V. Vapnik and A. Chervonenkis. *Theory of Pattern Recognition [in Russian]*. Nauka, Moscow, 1974. (German Translation: W. Wapnik & A. Tscherwonenkis, *Theorie der Zeichenerkennung*, Akademie–Verlag, Berlin, 1979).

[6] R. M. Dudley. A course on empirical processes. *Lecture Notes in Mathematics*, 1097:2–142, 1984.

[7] L. Devroye and G. Lugosi. *Combinatorial Methods in Density Estimation*. Springer Series in Statistics. Springer Verlag, New York, 2001.

[8] P. Bartlett, O. Bousquet, and S. Mendelson. Local rademacher complexities. Preprint, 2003.

[9] D. A. McAllester. Simplified pac-bayesian margin bounds. In *Proceedings of Computational Learning Theory (COLT)*, 2003.

[10] M. Ledoux and M. Talagrand. *Probability in Banach spaces*. Springer-Verlag, Berlin, 1991.

[11] M. Talagrand. The Glivenko-Cantelli problem. *Annals of Probability*, 6:837–870, 1987.

[12] O. Catoni. Localized empirical complexity bounds and randomized estimators, 2003. Preprint.

[13] J.-Y. Audibert. Data-dependent generalization error bounds for (noisy) classification: a PAC-bayesian approach. 2003. Work in progress.
